# Temporal Coherence, Natural Image Sequences, and the Visual Cortex

**Jarmo Hurri** and **Aapo Hyvärinen**
Neural Networks Research Centre
Helsinki University of Technology
P.O.Box 9800, 02015 HUT, Finland
*{jarmo.hurri,aapo.hyvarinen}@hut.fi*

## Abstract

We show that two important properties of the primary visual cortex emerge when the principle of temporal coherence is applied to natural image sequences. The properties are simple-cell-like receptive fields and complex-cell-like pooling of simple cell outputs, which emerge when we apply two different approaches to temporal coherence. In the first approach we extract receptive fields whose outputs are as temporally coherent as possible. This approach yields simple-cell-like receptive fields (oriented, localized, multiscale). Thus, temporal coherence is an alternative to sparse coding in modeling the emergence of simple cell receptive fields. The second approach is based on a two-layer statistical generative model of natural image sequences. In addition to modeling the temporal coherence of individual simple cells, this model includes inter-cell temporal dependencies. Estimation of this model from natural data yields both simple-cell-like receptive fields, and complex-cell-like pooling of simple cell outputs. In this completely unsupervised learning, both layers of the generative model are estimated simultaneously from scratch. This is a significant improvement on earlier statistical models of early vision, where only one layer has been learned, and others have been fixed a priori.

## 1 Introduction

The functional role of simple and complex cells has puzzled scientists since their response properties were first mapped by Hubel and Wiesel in the 1950s (see, e.g., [1]). The current view of the functionality of sensory neural networks emphasizes learning and the relationship between the structure of the cells and the statistical properties of the information they process (see, e.g., [2]). In 1996 a major advance was achieved when Olshausen and Field showed that simple-cell-like receptive fields emerge when sparse coding is applied to natural image data [3]. Similar results were obtained with independent component analysis shortly thereafter [4]. In the case of image data, independent component analysis is closely related to sparse coding [5].

In this paper we show that a principle called *temporal coherence* [6, 7, 8, 9] leads to the emergence of major properties of the primary visual cortex from natural image sequences.

Temporal coherence is based on the idea that when processing temporal input, the representation changes as little as possible over time. Several authors have demonstrated the usefulness of this principle using simulated data (see, e.g., [6, 7]).

We apply the principle of temporal coherence to natural input, and at the level of early vision, in two different ways. In the first approach we show that when the input consists of natural image sequences, the maximization of temporal response strength correlation of cell output leads to receptive fields which are similar to simple cell receptive fields. These results show that temporal coherence is an alternative to sparse coding, in that they both result in the emergence of simple-cell-like receptive fields from natural input data. Whereas earlier research has focused on establishing a link between temporal coherence and complex cells, our results demonstrate that such a connection exists even on the simple cell level. We will also show how this approach can be interpreted as estimation of a linear latent variable model in which the latent signals have varying variances.

In the second approach we use the principle of temporal coherence to formulate a two-layer generative model of natural image sequences. In addition to single-cell temporal coherence, this model also captures inter-cell temporal dependencies. We show that when this model is estimated from natural image sequence data, the results include both simple-cell-like receptive fields, and a complex-cell-like pooling of simple cell outputs. Whereas in earlier research learning two-layer statistical models of early vision has required fixing one of the layers beforehand, in our model both layers are learned simultaneously.

## 2 Simple-cell-like receptive fields are temporally coherent features

Our first approach to modeling temporal coherence in natural image sequences can be interpreted either as maximization of temporal coherence of cell outputs, or as estimation of a latent variable model in which the underlying variables have certain kind of time structure. This situation is analogous to sparse coding, because measures of sparseness can also be used to estimate linear generative models with non-Gaussian independent sources [5]. We first describe our measure of temporal coherence, and then provide the link to latent variable models.

In this paper we restrict ourselves to consider linear spatial models of simple cells. Linear simple cell models are commonly used in studies concerning the connections between visual input statistics and simple cell receptive fields [3, 4]. (Non-negative and spatiotemporal extensions of this basic framework are discussed in [10].) The linear spatial model uses a set of spatial filters (vectors) $\mathbf{w}_1, ..., \mathbf{w}_K$ to relate input to output. Let signal vector $\mathbf{x}(t)$ denote the input of the system at time $t$. A vectorization of image patches can be done by scanning images column-wise into vectors – for windows of size $N \times N$ this yields vectors with dimension $N^2$. The output of the $k$th filter at time $t$, denoted by signal $y_k(t)$, is given by $y_k(t) = \mathbf{w}_k^T \mathbf{x}(t)$. Let matrix $\mathbf{W} = [\mathbf{w}_1 \cdots \mathbf{w}_K]^T$ denote a matrix with all the filters as rows. Then the input-output relationship can be expressed in vector form by

$$\mathbf{y}(t) = \mathbf{W}\mathbf{x}(t), \tag{1}$$

where signal vector $\mathbf{y}(t) = [y_1(t) \cdots y_K(t)]^T$.

Temporal response strength correlation, the objective function, is defined by

$$f(\mathbf{W}) = \sum_{k=1}^{K} \mathrm{E}_t \left\{ g(y_k(t)) g(y_k(t - \Delta t)) \right\}, \tag{2}$$

where the nonlinearity $g$ is strictly convex, even (rectifying), and differentiable. The symbol $\Delta t$ denotes a delay in time. The nonlinearity $g$ measures the strength (amplitude) of the response of the filter, and emphasizes large responses over small ones (see [10] for

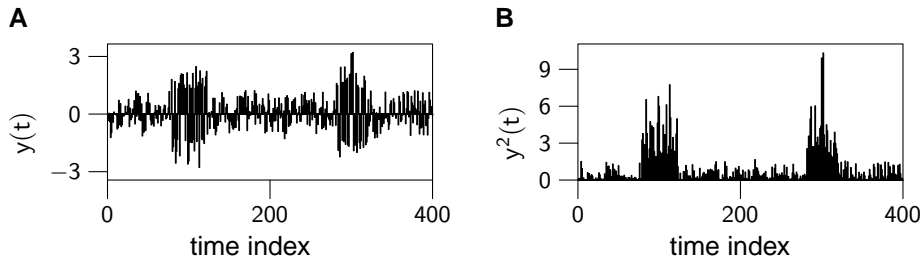

Figure 1: Illustration of nonstationarity of variance. (A) A temporally uncorrelated signal $y(t)$ with nonstationary variance. (B) Plot of $y^2(t)$.

additional discussion). Examples of choices for this nonlinearity are $g_1(\alpha) = \alpha^2$, which measures the energy of the response, and $g_2(\alpha) = \ln \cosh \alpha$, which is a robustified version of $g_1$. A set of filters which has a large temporal response strength correlation is such that the same filters *often respond strongly at consecutive time points*, outputting large (either positive or negative) values. This means that the same filters will respond strongly over short periods of time, thereby expressing temporal coherence of a population code. A detailed discussion of the difference between temporal response strength correlation and sparseness, including several control experiments, can be found in [10].

To keep the outputs of the filters bounded we enforce the unit variance constraint on each of the output signals $y_k(t)$. Additional constraints are needed to keep the filters from converging to the same solution – we force their outputs to be uncorrelated. A gradient projection method can be used to maximize (2) under these constraints. The initial value of $\mathbf{W}$ is selected randomly. See [10] for details.

The interpretation of maximization of objective function (2) as estimation of a generative model is based on the concept of sources with nonstationary variances [11, 12]. The linear generative model for $\mathbf{x}(t)$, the counterpart of equation (1), is similar to the one in [13, 3]:

$$\mathbf{x}(t) = \mathbf{A}\mathbf{y}(t). \tag{3}$$

Here $\mathbf{A} = [\mathbf{a}_1 \cdots \mathbf{a}_K]$ denotes a matrix which relates the image patch $\mathbf{x}(t)$ to the activities of the simple cells, so that each column $\mathbf{a}_k$, $k = 1, ..., K$, gives the feature that is coded by the corresponding simple cell. The dimension of $\mathbf{x}(t)$ is typically larger than the dimension of $\mathbf{y}(t)$, so that (1) is generally not invertible but an underdetermined set of linear equations. A one-to-one correspondence between $\mathbf{W}$ and $\mathbf{A}$ can be established by computing the pseudoinverse solution $\mathbf{A} = \mathbf{W}^T(\mathbf{W}\mathbf{W}^T)^{-1}$.

The nonstationarity of the variances of sources $\mathbf{y}(t)$ means that their variances change over time, and the variance of a signal is correlated at nearby time points. An example of a signal with nonstationary variance is shown in Figure 1. It can be shown [12] that optimization of a cumulant-based criterion, similar to equation (2), can separate independent sources with nonstationary variances. Thus, the maximization of the objective function can also be interpreted as estimation of generative models in which the activity levels of the sources vary over time, and are temporally correlated over time. As was noted above, this situation is analogous to the application of measures of sparseness to estimate linear generative models with non-Gaussian sources.

The algorithm was applied to natural image sequence data, which was sampled from a subset of image sequences used in [14]. The number of samples was 200,000, $\Delta t$ was 40 ms, and the sampled image patches were of size $16 \times 16$ pixels. Preprocessing consisted of temporal decorrelation, subtraction of local mean, and normalization [10], and dimensionality reduction from 256 to 160 using principal component analysis [5] (this degree of reduction

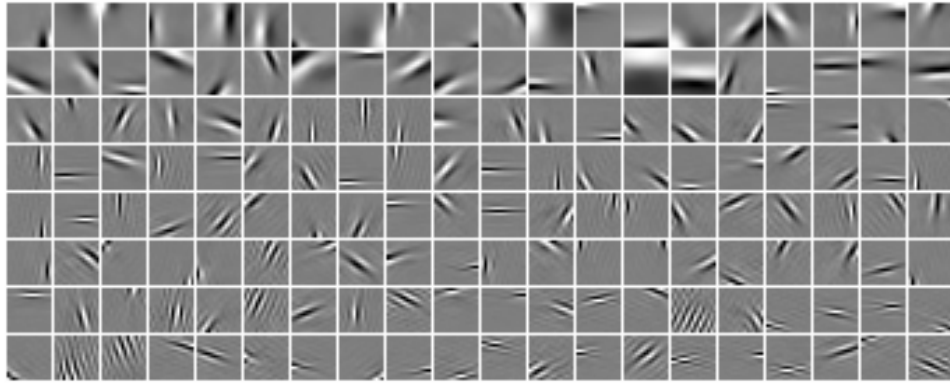

Figure 2: Basis vectors estimated using the principle of temporal coherence. The vectors were estimated from natural image sequences by optimizing temporal response strength correlation (2) under unit energy and uncorrelatedness constraints (here non-linearity $g(\alpha) = \ln \cosh \alpha$). The basis vectors have been ordered according to $E_t \left\{ g(y_k(t)) g(y_k(t - \Delta t)) \right\}$, that is, according to their "contribution" into the final objective value (vectors with largest values top left).

retains 95% of signal energy).

Figure 2 shows the basis vectors (columns of matrix $\mathbf{A}$) which emerge when temporal response strength correlation is maximized for this data. The basis vectors are oriented, localized, and have multiple scales. These are the main features of simple cell receptive fields [1]. A quantitative analysis, showing that the resulting receptive fields are similar to those obtained using sparse coding, can be found in [10], where the details of the experiments are also described.

## 3 Inter-cell temporal dependencies yield simple cell output pooling

### 3.1 Model

Temporal response strength correlation, equation (2), measures the temporal coherence of individual simple cells. In terms of the generative model described above, this means that the nonstationary variances of different $y_k(t)$'s have no interdependencies. In this section we add another layer to the generative model presented above to extend the theory to simple cell interactions, and to the level of complex cells.

Like in the generative model described at the end of the previous section, the output layer of the model (see Figure 3) is linear, and maps signed cell responses to image features. But in contrast to the previous section, or models used in independent component analysis [5] or basic sparse coding [3], we do *not* assume that the components of $\mathbf{y}(t)$ are independent. Instead, we model the dependencies between these components with a multivariate autoregressive model in the first layer of our model. Let $\mathbf{abs}\,(\mathbf{y}(t)) = [|y_1(t)| \cdots |y_K(t)|]^T$, let $\mathbf{v}(t)$ denote a driving noise signal, and let $\mathbf{M}$ denote a $K \times K$ matrix. Our model is a *multidimensional first-order autoregressive process*, defined by

$$\mathbf{abs}\,(\mathbf{y}(t)) = \mathbf{M}\,\mathbf{abs}\,(\mathbf{y}(t - \Delta t)) + \mathbf{v}(t). \tag{4}$$

As in independent component analysis, we also need to fix the scale of the latent variables by defining $E_t \left\{ y_k^2(t) \right\} = 1$ for $k = 1, ..., K$.

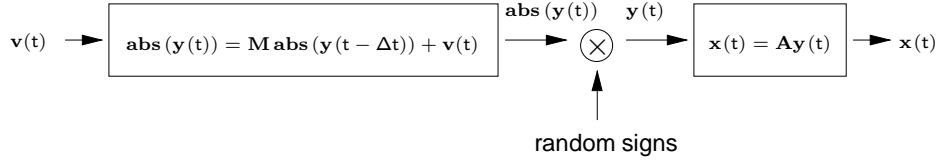

Figure 3: The two layers of the generative model. Let $\mathbf{abs}\,(\mathbf{y}\,(t)) = [|y_1(t)| \cdots |y_K(t)|]^T$ denote the amplitudes of simple cell responses. In the first layer, the driving noise signal $\mathbf{v}(t)$ generates the amplitudes of simple cell responses via an autoregressive model. The signs of the responses are generated randomly between the first and second layer to yield signed responses $\mathbf{y}(t)$. In the second layer, natural video $\mathbf{x}(t)$ is generated linearly from simple cell responses. In addition to the relations shown here, the generation of $\mathbf{v}(t)$ is affected by $\mathbf{M}\,\mathbf{abs}\,(\mathbf{y}\,(t - \Delta t))$ to ensure non-negativity of $\mathbf{abs}\,(\mathbf{y}\,(t))$. See text for details.

There are dependencies between the driving noise $\mathbf{v}(t)$ and output strengths $\mathbf{abs}\,(\mathbf{y}\,(t))$, caused by the non-negativity of $\mathbf{abs}\,(\mathbf{y}\,(t))$. To take these dependencies into account, we use the following formalism. Let $\mathbf{u}(t)$ denote a random vector with components which are statistically independent of each other. We define $\mathbf{v}(t) = \mathbf{max}\,(-\mathbf{M}\,\mathbf{abs}\,(\mathbf{y}\,(t - \Delta t))\,, \mathbf{u}(t))$, where, for vectors $\mathbf{a}$ and $\mathbf{b}$, $\mathbf{max}\,(\mathbf{a}, \mathbf{b}) = [\max(a_1, b_1) \cdots \max(a_n, b_n)]^T$. We assume that $\mathbf{u}(t)$ and $\mathbf{abs}\,(\mathbf{y}\,(t))$ are uncorrelated.

To make the generative model complete, a mechanism for generating the signs of cell responses $\mathbf{y}(t)$ must be included. We specify that the signs are generated randomly with equal probability for plus or minus after the strengths of the responses have been generated. Note that one consequence of this is that the different $y_k(t)$'s are uncorrelated. In the estimation of the model this uncorrelatedness property is used as a constraint. When this is combined with the unit variance (scale) constraints described above, the resulting set of constraints is the same as in the approach described in Section 2.

In equation (4), a large positive matrix element $\mathbf{M}(i, j)$, or $\mathbf{M}(j, i)$, indicates that there is strong temporal coherence between the output strengths of cells $i$ and $j$. Thinking in terms of grouping temporally coherent cells together, matrix $\mathbf{M}$ can be thought of as containing similarities (reciprocals of distances) between different cells. We will use this property in the experimental section to derive a topography of simple cell receptive fields from $\mathbf{M}$.

### 3.2 Estimation of the model

To estimate the model defined above we need to estimate both $\mathbf{M}$ and $\mathbf{W}$ (pseudoinverse of $\mathbf{A}$). We first show how to estimate $\mathbf{M}$, given $\mathbf{W}$. We then describe an objective function which can be used to estimate $\mathbf{W}$, given $\mathbf{M}$. Each iteration of the estimation algorithm consists of two steps. During the first step $\mathbf{M}$ is updated, and $\mathbf{W}$ is kept constant; during the second step these roles are reversed.

First, regarding the estimation of $\mathbf{M}$, consider a situation in which $\mathbf{W}$ is kept constant. It can be shown that $\mathbf{M}$ can be estimated by using approximative method of moments, and that the estimate is given by

$$\mathbf{M} \approx \beta \mathrm{E}_t \left\{ (\mathbf{abs}\,(\mathbf{y}\,(t)) - \mathrm{E}_t \,\{\mathbf{abs}\,(\mathbf{y}\,(t))\}) \,(\mathbf{abs}\,(\mathbf{y}\,(t - \Delta t)) - \mathrm{E}_t \,\{\mathbf{abs}\,(\mathbf{y}\,(t))\})^T \right\}$$
$$\times \, \mathrm{E}_t \left\{ (\mathbf{abs}\,(\mathbf{y}\,(t)) - \mathrm{E}_t \,\{\mathbf{abs}\,(\mathbf{y}\,(t))\}) \,(\mathbf{abs}\,(\mathbf{y}\,(t)) - \mathrm{E}_t \,\{\mathbf{abs}\,(\mathbf{y}\,(t))\})^T \right\}^{-1},$$
$$(5)$$

where $\beta > 1$. Since this multiplier has a constant linear effect in the objective function

given below, its value does not change the optima, so we can set $\beta = 1$ in the optimization. (Details are given in [15].) The resulting estimator is the same as the optimal least mean squares linear predictor in the case of unconstrained $\mathbf{v}(t)$.

The estimation of $\mathbf{W}$ is more complicated. A rigorous derivation of an objective function based on well-known estimation principles is very difficult, because the statistics involved are non-Gaussian, and the processes have difficult interdependencies. Therefore, instead of deriving an objective function from first principles, we derived an objective function heuristically, and verified through simulations that the objective function is capable of estimating the two-layer model. The objective function is a weighted sum of the covariances of filter output strengths at times $t - \Delta t$ and $t$, defined by

$$f(\mathbf{W}, \mathbf{M}) = \sum_{i=1}^{K} \sum_{j=1}^{K} \mathbf{M}(i,j) \operatorname{cov}\left\{ |y_i(t)|, |y_j(t - \Delta t)| \right\}. \tag{6}$$

In the actual estimation algorithm, $\mathbf{W}$ is updated by employing a gradient projection approach to the optimization of (6) under the constraints. The initial value of $\mathbf{W}$ is selected randomly.

The fact that the algorithm described above is able to estimate the two-layer model has been verified through extensive simulations (details can be found in [15]).

### 3.3 Experiments

The estimation algorithm was run on the same data set as in the previous experiment (see Section 2). The extracted matrices $\mathbf{A}$ and $\mathbf{M}$ can be visualized simultaneously by using the interpretation of $\mathbf{M}$ as a similarity matrix (see Section 3.1). Figure 4 illustrates the basis vectors – that is, columns of $\mathbf{A}$ – laid out at spatial coordinates derived from $\mathbf{M}$ in a way explained below. The resulting basis vectors are again oriented, localized and multiscale, as in the previous experiment.

The two-dimensional coordinates of the basis vectors were determined from $\mathbf{M}$ using multidimensional scaling (see figure caption for details). The temporal coherence between the outputs of two cells $i$ and $j$ is reflected in the distance between the corresponding receptive fields: the larger the elements $\mathbf{M}(i,j)$ and $\mathbf{M}(j,i)$ are, the closer the receptive fields are to each other. We can see that local topography emerges in the results: those basis vectors which are close to each other seem to be mostly coding for similarly oriented features at nearby spatial positions. This kind of grouping is characteristic of pooling of simple cell outputs at complex cell level [1].[1]

Thus, the estimation of our two-layer model from natural image sequences yields both simple-cell-like receptive fields, and grouping similar to the pooling of simple cell outputs. Linear receptive fields emerge in the second layer (matrix $\mathbf{A}$), and cell output grouping emerges in the first layer (matrix $\mathbf{M}$). Both of these layers are estimated simultaneously. This is a significant improvement on earlier statistical models of early vision, because no a priori fixing of either of these layers is needed.

## 4 Conclusions

We have shown in this paper that when the principle of temporal coherence is applied to natural image sequences, both simple-cell-like receptive fields, and complex-cell-like pooling of simple cell outputs emerge. These results were obtained with two different approaches

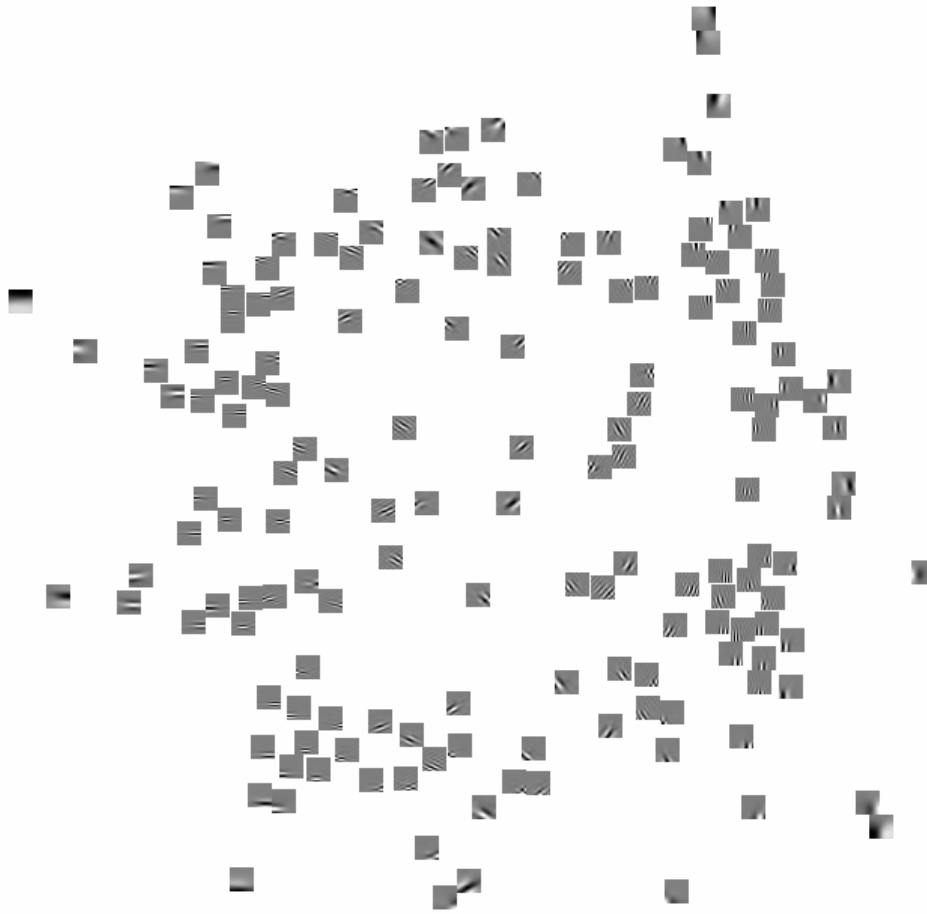

Figure 4: Results of estimating the two-layer generative model from natural image sequences. Basis vectors (columns of **A**) plotted at spatial coordinates given by applying multidimensional scaling to **M**. Matrix **M** was first converted to a non-negative similarity matrix $\mathbf{M}_s$ by subtracting $\min_{i,j} \mathbf{M}(i,j)$ from each of its elements, and by setting each of the diagonal elements at value 1. Multidimensional scaling was then applied to $\mathbf{M}_s$ by interpreting entries $\mathbf{M}_s(i,j)$ and $\mathbf{M}_s(j,i)$ as similarity measures between cells $i$ and $j$. Some of the resulting coordinates were very close to each other, so tight cell clusters were magnified for purposes of visual display. Details are given in [15].

to temporal coherence. The first used temporally coherent simple cell outputs, and the second was based on a temporal two-layer generative model of natural image sequences. Simple-cell-like receptive fields emerge in both cases, and the output pooling emerges as a local topographic property in the case of the two-layer generative model.

These results are important for two reasons. First, to our knowledge this is the first time that localized and oriented receptive fields with different scales have been shown to emerge from natural data using the principle of temporal coherence. In some models of invariant visual representations [8, 16] simple cell receptive fields are obtained as by-products, but learning is strongly modulated by complex cells, and the receptive fields seem to lack the important properties of spatial localization and multiresolution. Second, in earlier research on statistical models of early vision, learning two-layer models has required a priori fixing of one of the layers. This is not needed in our two-layer model, because both layers emerge simultaneously in a completely unsupervised manner from the natural input data.

## Footnotes

[1]Some global topography also emerges: those basis vectors which code for horizontal features are on the left in the figure, while those that code for vertical features are on the right.

## References

[1] Stephen E. Palmer. *Vision Science – Photons to Phenomenology*. The MIT Press, 1999.

[2] Eero P. Simoncelli and Bruno A. Olshausen. Natural image statistics and neural representation. *Annual Review of Neuroscience*, 24:1193–1216, 2001.

[3] Bruno A. Olshausen and David Field. Emergence of simple-cell receptive field properties by learning a sparse code for natural images. *Nature*, 381(6583):607–609, 1996.

[4] Anthony Bell and Terrence J. Sejnowski. The independent components of natural scenes are edge filters. *Vision Research*, 37(23):3327–3338, 1997.

[5] Aapo Hyvärinen, Juha Karhunen, and Erkki Oja. *Independent Component Analysis*. John Wiley & Sons, 2001.

[6] Peter Földiák. Learning invariance from transformation sequences. *Neural Computation*, 3(2):194–200, 1991.

[7] James Stone. Learning visual parameters using spatiotemporal smoothness constraints. *Neural Computation*, 8(7):1463–1492, 1996.

[8] Christoph Kayser, Wolfgang Einhäuser, Olaf Dümmer, Peter König, and Konrad Körding. Extracting slow subspaces from natural videos leads to complex cells. In Georg Dorffner, Horst Bischof, and Kurt Hornik, editors, *Artificial Neural Networks – ICANN 2001*, volume 2130 of *Lecture notes in computer science*, pages 1075–1080. Springer, 2001.

[9] Laurenz Wiskott and Terrence J. Sejnowski. Slow feature analysis: Unsupervised learning of invariances. *Neural Computation*, 14(4):715–770, 2002.

[10] Jarmo Hurri and Aapo Hyvärinen. Simple-cell-like receptive fields maximize temporal coherence in natural video. *Neural Computation*, 2003. In press.

[11] Kiyotoshi Matsuoka, Masahiro Ohya, and Mitsuru Kawamoto. A neural net for blind separation of nonstationary signals. *Neural Networks*, 8(3):411–419, 1995.

[12] Aapo Hyvärinen. Blind source separation by nonstationarity of variance: A cumulant-based approach. *IEEE Transactions on Neural Networks*, 12(6):1471–1474, 2001.

[13] Aapo Hyvärinen and Patrik O. Hoyer. A two-layer sparse coding model learns simple and complex cell receptive fields and topography from natural images. *Vision Research*, 41(18):2413–2423, 2001.

[14] J. Hans van Hateren and Dan L. Ruderman. Independent component analysis of natural image sequences yields spatio-temporal filters similar to simple cells in primary visual cortex. *Proceedings of the Royal Society of London B*, 265(1412):2315–2320, 1998.

[15] Jarmo Hurri and Aapo Hyvärinen. A two-layer dynamic generative model of natural image sequences. Submitted.

[16] Teuvo Kohonen, Samuel Kaski, and Harri Lappalainen. Self-organized formation of various invariant-feature filters in the adaptive-subspace SOM. *Neural Computation*, 9(6):1321–1344, 1997.
